# Tracking Dynamic Sources of Malicious Activity at Internet-Scale

**Shobha Venkataraman**[*]**, Avrim Blum**[†]**, Dawn Song**[◇]**, Subhabrata Sen**[*]**, Oliver Spatscheck**[*]
[*] AT&T Labs – Research   {shvenk,sen,spatsch}@research.att.com
[†]Carnegie Mellon University   avrim@cs.cmu.edu
[◇]University of California, Berkeley   dawnsong@cs.berkeley.edu

## Abstract

We formulate and address the problem of discovering dynamic malicious regions on the Internet. We model this problem as one of adaptively pruning a known decision tree, but with additional challenges: (1) severe space requirements, since the underlying decision tree has over 4 billion leaves, and (2) a changing target function, since malicious activity on the Internet is dynamic. We present a novel algorithm that addresses this problem, by putting together a number of different "experts" algorithms and online paging algorithms. We prove guarantees on our algorithm's performance as a function of the best possible pruning of a similar size, and our experiments show that our algorithm achieves high accuracy on large real-world data sets, with significant improvements over existing approaches.

## 1   Introduction

It is widely acknowledged that identifying the regions that originate malicious traffic on the Internet is vital to network security and management, e.g., in throttling attack traffic for fast mitigation, isolating infected sub-networks, and predicting future attacks [6, 18, 19, 24, 26]. In this paper, we show how this problem can be modeled as a version of a question studied by Helmbold and Schapire [11] of adaptively learning a good pruning of a known decision tree, but with a number of additional challenges and difficulties. These include a changing target function and severe space requirements due to the enormity of the underlying IP address-space tree. We develop new algorithms able to address these difficulties that combine the underlying approach of [11] with the sleeping experts framework of [4, 10] and the online paging problem of [20]. We show how to deal with a number of practical issues that arise and demonstrate empirically on real-world datasets that this method substantially improves over existing approaches of /24 prefixes and network-aware clusters [6, 19, 24] in correctly identifying malicious traffic. Our experiments on data sets of 126 million IP addresses demonstrate that our algorithm is able to achieve a clustering that is both highly accurate and meaningful.

### 1.1   Background

Multiple measurement studies have indicated that malicious traffic tends to cluster in a way that aligns with the structure of the IP address space, and that this is true for many different kinds of malicious traffic – spam, scanning, botnets, and phishing [6, 18, 19, 24]. Such clustered behaviour can be easily explained: most malicious traffic originates from hosts in poorly-managed networks, and networks are typically assigned contiguous blocks of the IP address space. Thus, it is natural that malicious traffic is clustered in parts of the IP address space that belong to poorly-managed networks.

From a machine learning perspective, the problem of identifying regions of malicious activity can be viewed as one of finding a good pruning of a known decision tree – the IP address space may be naturally interpreted as a binary tree (see Fig.1(a)), and the goal is to learn a pruning of this tree that is not too large and has low error in classifying IP addresses as malicious or non-malicious. The structure of the IP address space suggests that there may well be a pruning with only a modest number of leaves that can classify most of the traffic accurately. Thus, identifying regions of malicious activity from an online stream of labeled data is much like the problem considered by Helmbold and Schapire [11] of adaptively learning a good pruning of a known decision tree. However, there are a

number of real-world challenges, both conceptual and practical, that must be addressed in order to make this successful.

One major challenge in our application comes from the scale of the data and size of a complete decision tree over the IP address space. A full decision tree over the IPv4 address space would have $2^{32}$ leaves, and over the IPv6 address space (which is slowly being rolled out), $2^{128}$ leaves. With such large decision trees, it is critical to have algorithms that do not build the complete tree, but instead operate in space comparable to the size of a good pruning. These space constraints are also important because of the volume of traffic that may need to be analyzed – ISPs often collect terabytes of data daily and an algorithm that needs to store all its data in memory simultaneously would be infeasible.

A second challenge comes from the fact that the regions of malicious activity may shift longitudinally over time [25]. This may happen for many reasons, e.g., administrators may eventually discover and clean up already infected bots, and attackers may target new vulnerabilities and attack new hosts elsewhere. Such dynamic behaviour is a primary reason why individual IP addresses tend to be such poor indicators of future malicious traffic [15, 26]. Thus, we cannot assume that the data comes from a fixed distribution over the IP address space; the algorithm needs to adapt to dynamic nature of the malicious activity, and track these changes accurately and quickly. That is, we must consider not only an online sequence of examples but also a changing target function.

While there have been a number of measurement studies [6, 18, 19, 24] that have examined the origin of malicious traffic from IP address blocks that are kept fixed apriori, none of these have focused on developing online algorithms that find the best predictive IP address tree. Our challenge is to develop an efficient high-accuracy online algorithm that handles the severe space constraints inherent in this problem and accounts for the dynamically changing nature of malicious behavior. We show that we can indeed do this, both proving theoretical guarantees on adaptive regret and demonstrating successful performance on real-world data.

### 1.2 Contributions
In this paper, we formulate and address the problem of discovering and tracking malicious regions of the IP address space from an online stream of data. We present an algorithm that adaptively prunes the IP address tree in a way that maintains at most $m$ leaves and performs nearly as well as the optimum adaptive pruning of the IP address tree with a comparable size. Intuitively, we achieve the required adaptivity and the space constraints by combining several "experts" algorithms together with a tree-based version of paging. Our theoretical results prove that our algorithm can predict nearly as well as the best adaptive decision tree with $k$ leaves when using $O(k \log k)$ leaves.

Our experimental results demonstrate that our algorithm identifies malicious regions of the IP address space accurately, with orders of magnitude improvement over previous approaches. Our experiments focus on classifying spammers and legitimate senders on two mail data sets, one with 126 million messages collected over 38 days from the mail servers of a tier-1 ISP, and a second with 28 million messages collected over 6 months from an enterprise mail server. Our experiments also highlight the importance of allowing the IP address tree to be dynamic, and the resulting view of the IP address space that we get is both compelling and meaningful.

## 2 Definitions and Preliminaries
We now present some basic definitions as well as our formal problem statement.

The IP address hierarchy can be naturally interpreted as a full binary tree, as in Fig. 1: the leaves of the tree correspond to individual IP addresses, and the non-leaf nodes correspond to the remaining IP prefixes. Let $\mathcal{P}$ denote the set of all IP prefixes, and $\mathcal{I}$ denote the set of all IP addresses. We also use term *clusters* to denote the IP prefixes.

We define an *IPTree* $T_P$ to be a pruning of the full IP address tree: a tree whose nodes are IP prefixes $P \in \mathcal{P}$, and whose leaves are each associated with a label, i.e., malicious or non-malicious. An IPtree can thus be interpreted as a classification function for the IP addresses $\mathcal{I}$: an IP address $i$ gets the label associated with its longest matching prefix in $P$. Fig. 1 shows an example of an IPtree. We define the *size* of an IPtree to be the number of leaves it has. For example, in Fig. 1(a), the size of the IPtree is 6.

As described in Sec. 1, we focus on online learning in this paper. A typical point of comparison used in the online learning model is the error of the *optimal offline fixed* algorithm. In this case, the optimal offline fixed algorithm is the IPtree of a given size $k$ i.e., the tree of size $k$ that makes

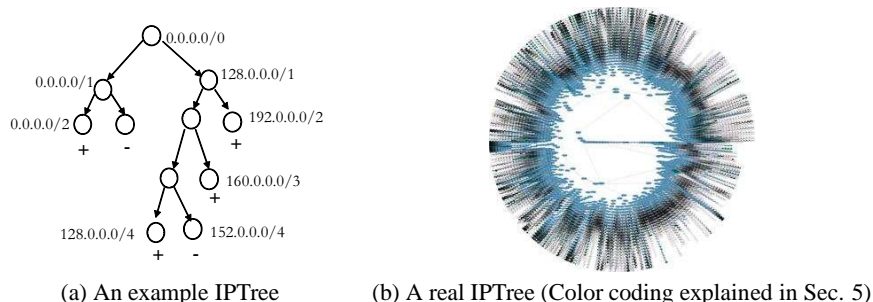

(a) An example IPTree      (b) A real IPTree (Color coding explained in Sec. 5)

Figure 1: IPTrees: example and real. Recall that an IP address is interpreted as a 32-bit string, read from left to right. This defines a path on the binary tree, going left for 0 and right for 1. An IP prefix is denoted by $\text{IP}/n$, where $n$ indicates the number of bits relevant to the prefix.

the fewest mistakes on the entire sequence. However, if the true underlying IPtree may change over time, a better point of comparison would allow the offline tree to also change over time. To make such a comparison meaningful, the offline tree must pay an additional penalty each time it changes (otherwise the offline tree would not be a meaningful point of comparison – it could change for each IP address in the sequence, and thus make no mistakes). We therefore limit the kinds of changes the offline tree can make, and compare the performance of our algorithm to every IPtree with $k$ leaves, as a function of the errors it makes and the changes it makes.

We define an *adaptive IPtree* of size $k$ to be an adaptive tree that can (a) grow nodes over time so long as it never has more than $k$ leaves, (b) change the labels of its leaf nodes, and (c) occasionally reconfigure itself completely. Our goal is to develop an online algorithm $T$ such that for any sequence of IP addresses, (1) for *every* adaptive tree $T'$ of size $k$, the number of mistakes made by $T$ is bounded by a (small) function of the mistakes and the changes of types (a), (b), and (c) made by $T'$, and (2) $T$ uses no more than $\tilde{O}(k)$ space. In the next section, we describe an algorithm meeting these requirements.

## 3    Algorithms and Analysis

In this section, we describe our main algorithm TrackIPTree, and present theoretical guarantees on its performance. At a high-level, our approach keeps a number of experts in each prefix of the IPtree, and combines their predictions to classify every IP address. The inherent structure in the IPtree allows us to decompose the problem into a number of expert problems, and provide lower memory bounds and better guarantees than earlier approaches.

We begin with an overview. Define the *path-nodes* of an IP address to be the set of all prefixes of $i$ in $T$, and denote this set by $P_{i,T}$. To predict the label of an IP $i$, the algorithm looks up all the path-nodes in $P_{i,T}$, considers their predictions, and combines these predictions to produce a final label for $i$. To update the tree, the algorithm rewards the path-nodes that predicted correctly, penalizes the incorrect ones, and modifies the tree structure if necessary.

To fill out this overview, there are four technical questions that we need to address: (1) Of all the path-nodes in $P_{i,T}$, how do we learn the ones that are the most important? (2) How do we learn the correct label to predict at a particular path-node in $P_{i,T}$ (i.e., positive or negative)? (3) How do we grow the IPtree appropriately, ensuring that it grows primarily the prefixes needed to improve the classification accuracy? (4) How do we ensure that the size of the IPtree stays bounded by $m$? We address these questions by treating them as separate subproblems, and we show how they fit together to become the complete algorithm in Figure 3.1.

### 3.1    Subproblems of *TrackIPTree*

We now describe our algorithm in detail. Since our algorithm decomposes naturally into the four subproblems mentioned above, we focus on each subproblem separately to simplify the presentation. We use the following notation in our descriptions: Recall from Sec. 2 that $m$ is the maximum number of leaves allowed to our algorithm, $k$ is the size of the optimal offline tree, and $P_{i,T}$ denotes the set of path-nodes, i.e., the prefixes of IP $i$ in the current IPtree $T$.

**Relative Importance of the Path Nodes** First, we consider the problem of deciding which of the prefix nodes in the path $P_{i,T}$ is most important. We formulate this as a *sleeping experts problem* [4, 10]. We set an expert in each node, and call them the *path-node experts*, and for an IP $i$, we consider the set of path-node experts in $P_{i,T}$ to be the "awake" experts, and the rest to be "asleep". The

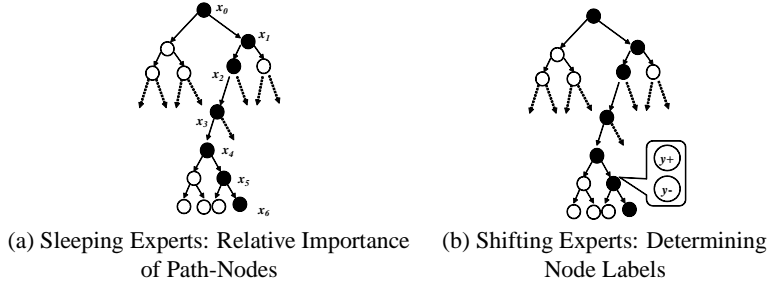

(a) Sleeping Experts: Relative Importance
of Path-Nodes

(b) Shifting Experts: Determining
Node Labels

Figure 2: Decomposing the TrackIPTree Algorithm

sleeping experts algorithm makes predictions using the awake experts, and intuitively, has the goal of predicting nearly as well as the best awake expert on the instance $i$ [1]. In our context, the best awake expert on the IP $i$ corresponds to the prefix of $i$ in the optimal IPtree, which remains sleeping until the IPtree grows that prefix. Fig. 2(a) illustrates the sleeping experts framework in our context: the shaded nodes are "awake" and the rest are "asleep".

Specifically, let $x_t$ denote the weight of the path-node expert at node $t$, and let $S_{i,T} = \sum_{t \in P_{i,T}} x_t$. To predict on IP address $i$, the algorithm chooses the expert at node $t$ with probability $x_t/S_{i,T}$. To update, the algorithm penalizes all incorrect experts in $P_{i,T}$, reducing their weight $x_t$ to $\gamma x_t$. (e.g., $\gamma = 0.8$). It then renormalizes the weights of all the experts in $P_{i,T}$ so that their sum $S_{i,T}$ does not change. (In our proof, we use a slightly different version of the sleeping experts algorithm [4]).

**Deciding Labels of Individual Nodes** Next, we need to decide whether the path-node expert at a node $n$ should predict positive or negative. We use a different experts algorithm to address this subproblem – the *shifting experts* algorithm [12]. Specifically, we allow each node $n$ to have two additional experts – a positive expert, which always predicts positive, and a negative expert, which always predicts negative. We call these experts *node-label* experts.

Let $y_{n,+}$ and $y_{n,-}$ denote the weights of the positive and negative node-label experts respectively, with $y_{n,-} + y_{n,+} = 1$. The algorithm operates as follows: to predict, the node predicts positive with probability $y_{n,+}$ and negative with probability $y_{n,-}$. To update, when the node receives a label, it increases the weight of the correct node-label expert by $\epsilon$, and decreases the weight of the incorrect node-label expert by $\epsilon$ (upto a maximum of 1 and a minimum of 0). Note that this algorithm naturally adapts when a leaf of the optimal IPtree switches labels – the relevant node in our IPtree will slowly shift weights from the incorrect node-label expert to the correct one, making an expected $\frac{1}{\epsilon}$ mistakes in the process. Fig. 2(b) illustrates the shifting experts setting on an IPtree: each node has two experts, a positive and a negative. Fig. 3 shows how it fits in with the sleeping experts algorithm.

**Building Tree Structure** We next address the subproblem of building the appropriate structure for the IPtree. The intuition here is: when a node in the IPtree makes many mistakes, then either that node has a subtree in the optimal IPtree that separates the positive and negative instances, or the optimal IPtree must also make the same mistakes. Since TrackIPTree cannot distinguish between these two situations, it simply splits any node that makes sufficient mistakes. In particular, TrackIPTree starts with only the root node, and tracks the number of mistakes made at every node. Every time a leaf makes $\frac{1}{\epsilon}$ mistakes, TrackIPTree splits that leaf into its children, and instantiates and initializes the relevant path-node experts and node-label experts of the children. In effect, it is as if the path-node experts of the children had been asleep till this point, but will now be "awake" for the appropriate IP addresses.

TrackIPTree waits for $\frac{1}{\epsilon}$ mistakes at each node before growing it, so that there is a little resilence with noisy data – otherwise, it would split a node every time the optimal tree made a mistake, and the IPtree would grow very quickly. Note also that it naturally incorporates the optimal IPtree growing a leaf; our tree will grow the appropriate nodes when that leaf has made $\frac{1}{\epsilon}$ mistakes.

**Bounding Size of IPtree** Since TrackIPTree splits any node after it makes $\frac{1}{\epsilon}$ mistakes, it is likely that the IPtree it builds is split much farther than the optimal IPtree – TrackIPTree does not know when to stop growing a subtree, and it splits even if the same mistakes are made by the optimal IPtree. While this excessive splitting does not impact the predictions of the path-node experts or the node-label experts significantly, we still need to ensure that the IPtree built by our algorithm does not become too large.

TRACKIPTREE
Input: tree size $m$, learning rate $\epsilon$, penalty factor $\gamma$
**Initialize:**
    Set $T := root$
    InitializeNode($root$)

**Prediction Rule:** Given IP $i$
    //Select a node-label expert
    for $n \in P_{i,T}$
        flip coin of bias $y_{n,+}$
        if heads, $predict[n] := +$
        else $predict[n] := -$
    //Select a path-node expert
    rval $:= predict[n]$ with weight
    $x_n / \sum_{t \in P} x_t$
    Return rval

**Update Rule:** Given IP $i$, label $r$
    //Update node-label experts
    for $n \in P_{i,T}$
        for label $z \in \{+, -\}$
            if $z = r, y_{n,z} := y_{n,z} + \epsilon$
            else $y_{n,z} := y_{n,z} - \epsilon$

**Update Rule (Contd.):**
    //Update path-node experts
    $s := \sum_{n \in P_{t,T}} x_n$
    for $n \in P_{i,T}$
    if $predict[n] \neq r$,
        penalize $x_n := \gamma x_n$
        $mistakes[x_n] + +$
        if $mistakes[x_n] > 1/\epsilon$ and $n$
          is leaf, $GrowTree(n)$
    Renormalize $x_n := x_n \frac{s}{\sum_{j \in P_{i,T}} x_j}$

**sub** INITIALIZENODE
Input: node $t$
    $x_t := 1; y_{t,+} := y_{t,-} := 0.5$
    $mistakes[t] := 0$

**sub** GROWTREE
Input: leaf $l$
    if $size(T) \geq m$
        Select nodes $N$ to discard with
          paging algorithm
    Split leaf $l$ into children $lc, rc$.
    InitializeNode($lc$), InitializeNode($rc$)

Figure 3: The Complete TrackIPTree Algorithm

We do this by framing it as a paging problem [20]: consider each node in the IPtree to be a page, and the maximum allowed nodes in the IPtree to be the size of the cache. The offline IPtree, which has $k$ leaves, needs a cache of size $2k$. The IPtree built by our algorithm may have at most $m$ leaves (and thus, $2m$ nodes, since it is a binary tree), and so the size of its cache is $2m$ and the offline cache is $2k$. We may then select nodes to be discarded as if they were pages in the cache once the IPtree grows beyond $2m$ nodes; so, for example, we may choose the least recently used nodes in the IPtree, with LRU as the paging algorithm. Our analysis shows that setting $m = O(\frac{k}{\epsilon^2} \log \frac{k}{\epsilon})$ suffices, when TrackIPTree uses FLUSH-WHEN-FULL (FWF) as its paging algorithm – this is a simple paging algorithm that discards all the pages in the cache when the cache is full, and restarts with an empty cache. We use FWF here for a clean analysis, and especially since in simple paging models, many algorithms achieve no better guarantees [20]. For our experiments, we implement LRU, and our results show that this approach, while perhaps not sophisticated, still maintains an accurate predictive IPtree.

### 3.2 Analysis
In this section, we present theoretical guarantees on TrackIPTree's performance. We show our algorithm performs nearly as well as best adaptive $k$-IPtree, bounding the number of mistakes made by our algorithm as a function of the number of mistakes, number of labels changes and number of complete reconfigurations of the optimal such tree in hindsight.

**Theorem 3.1** *Fix $k$. Set the maximum number of leaves allowed to the TrackIPTree algorithm $m$ to be $\frac{10k}{\epsilon^2} \log \frac{k}{\epsilon}$. Let $T$ be an adaptive $k$-IPtree. Let $\Delta_{T,z}$ denote the number of times $T$ changes labels on the its leaves over the sequence $z$, and $R_{T,z}$ denote the number of times times $T$ has completely reconfigured itself over $z$.*

*The algorithm TrackIPTree ensures that on any sequence of instances $z$, for each $T$, the number of mistakes made by TrackIPTree is at most $(1 + 3\epsilon)M_{T,z} + (\frac{1}{\epsilon} + 3)\Delta_{T,z} + \frac{10k}{\epsilon^3} \log \frac{k}{\epsilon}(R_{T,z} + 1)$ with probability at least $1 - \left(\frac{1}{k}\right)^{\frac{k}{2\epsilon^2}}$.*

In other words, if there is an offline adaptive $k$-IPtree, that makes few changes and few mistakes on the input sequence of IP addresses, then TrackIPTree will also make only a small number of mistakes. Due to space constraints, we present the proof in the technical report [23].

## 4 Evaluation Setup
We now describe our evaluation set-up: data, practical changes to the algorithm, and baseline schemes that compare against. While there are many issues that go into converting the algorithm in Sec. 3 for practical use, we describe here those most important to our experiments, and defer the rest to the technical report [23].

**Data** We focus on IP addresses derived from mail data, since spammers represent a significant fraction of the malicious activity and compromised hosts on the Internet [6], and labels are relatively easy to obtain from spam-filtering run by the mail servers. For our evaluation, we consider labels from the mail servers' spam-filtering to be ground truth. Any errors in the spam-filtering will influence the tree that we construct and our experimental results are limited by this assumption.

One data set consists of log extracts collected at the mail servers of a tier-1 ISP with 1 million active mailboxes. The extracts contain the IP addresses of the mail servers that send mail to the ISP, the number of messages they sent, and the fraction of those messages that are classified as spam, aggregated over 10 minute intervals. The mail server's spam-filtering software consists of a combination of hand-crafted rules, DNS blacklists, and Brightmail [1], and we take their results as labels for our experiments. The log extracts were collected over 38 days from December 2008 to January 2009, and contain 126 million IP addresses, of which 105 million are spam and 21 million are legitimate.

The second data set consists of log extracts from the enterprise mail server of a large corporation with 1300 active mailboxes. These extracts also contain the IP addresses of mail servers that attempted to send mail, along with the number of messages they sent and the fraction of these messages that were classified spam by SpamAssassin [2], aggregated over 10 minute intervals. The extracts contain 28 million IP addresses, of which around 1.2 million are legitimate and the rest are spammers.

Note that in both cases, our data only contains aggregate information about the IP addresses of the mail servers *sending* mail to the ISP and enterprise mail servers, and so we do not have the ability to map any information back to individual users of the ISP or enterprise mail servers.

**TrackIPTree** For the experimental results, we use LRU as the paging algorithm when nodes need to be discarded from the IPtree (Sec. 3.1). In our implementation, we set TrackIPTree to discard 1% of $m$, the maximum leaves allowed, every time it needs to expire nodes. The learning rate $\epsilon$ is set to 0.05 and the penalty factor $\gamma$ for sleeping experts is set to 0.1 respectively. Our results are not affected if these parameters are changed by a factor of 2-3.

While we have presented an online learning algorithm, in practice, it will often need to predict on data without receiving labels of the instances right away. Therefore, we study TrackIPTree's accuracy on the following day's data, i.e., to compute prediction accuracy of day $i$, TrackIPTree is allowed to update until day $i-1$. We choose intervals of a day's length to allow the tree's predictions to be updated at least every day.

**Apriori Fixed Clusters** We compare TrackIPTree to two sets of *apriori fixed clusters*: (1) network-aware clusters, which are a set of unique prefixes derived from BGP routing table snapshots [17], and (2) /24 prefixes. We choose these clusters as a baseline, as they have been the basis of measurement studies discussed earlier (Sec. 1), prior work in IP-based classification [19, 24], and are even used by popular DNS blacklists [3].

We use the fixed clusters to predict the label of an IP in the usual manner: we simply assign an IP the label of its longest matching prefix among the clusters. Of course, we first need to assign these clusters their own labels. To ensure that they classify as well as possible, we assign them the optimal labeling over the data they need to classify; we do this by allowing them to make multiple passes over the data. That is, for each day, we assign labels so that the fixed clusters maximize their accuracy on spam for a given required accuracy on legitimate mail [2]. It is clear that this experimental set-up is favourable to the apriori fixed clusters.

We do not directly compare against the algorithm in [11], as it requires every unique IP address in the data set to be instantiated in the tree. In our experiments (e.g., with the ISP logs), this means that it requires over 90 million leaves in the tree. We instead focus on practical prior approaches with more cluster sizes in our experiments.

## 5 Results

We report three sets of experimental results regarding the prediction accuracy of TrackIPTree using the experimental set-up of Section 4. While we do not provide an extensive evaluation of our algorithm's computational efficiency, we note that our (unoptimized) implementation of TrackIPTree takes under a minute to learn over a million IP addresses, on a 2.4GHz Sparc64-VI core.

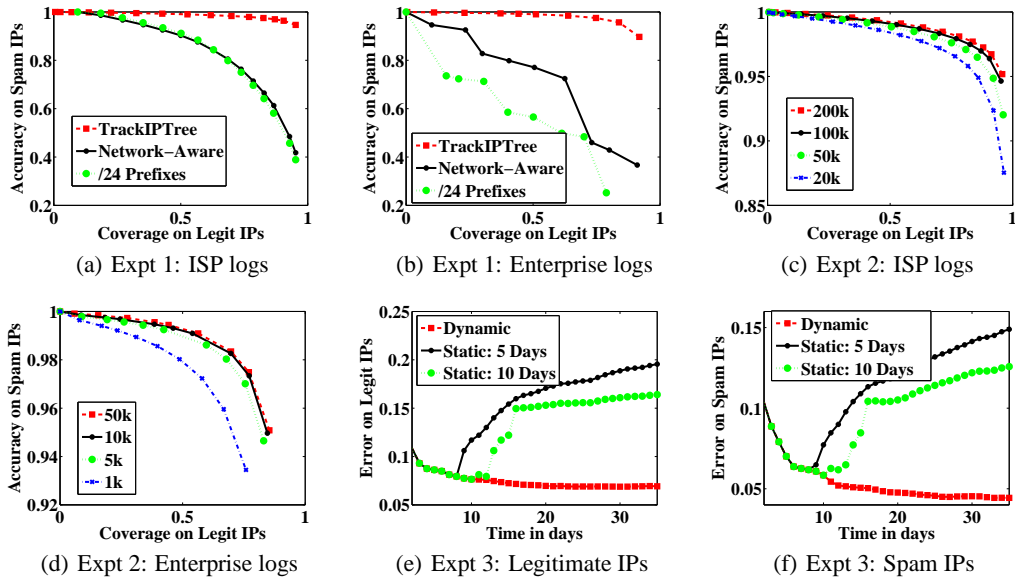

Figure 4: Results for Experiments 1, 2, and 3

Our results compare the fraction of spamming IPs that the clusters classify correctly, subject to the constraint that they classify at least $x\%$ legitimate mail IPs correctly (we term this to be the coverage of the legitimate IPs required). Thus, we effectively plot the true positive rate against the true negative rate. (This is just the ROC curve with the $x$-axis reversed, since we plot the true positive against the true negative, instead of plotting the true positive against the false positive.)

**Experiment 1: Comparisons with Apriori Fixed Clusters** Our first set of experiments compares the performance of our algorithm with network-aware clusters and /24 IP prefixes. Figs. 4(a) & 4(b) illustrate the accuracy tradeoff of the three sets of clusters on the two data sets. Clearly, the accuracy of TrackIPTree is a tremendous improvement on both sets of apriori fixed clusters – for any choice of coverage on legitimate IPs, the accuracy of spam IPs by TrackIPTree is far higher than the apriori fixed clusters, even by as much as a factor of 2.5. In particular, note that when the coverage required on legitimate IPs is $95\%$, TrackIPTree achieves $95\%$ accuracy in classifying spam on both data sets, compared to the $35 - 45\%$ achieved by the other clusters.

In addition, TrackIPTree gains this classification accuracy using a far smaller tree. Table 1 shows the median number of leaves instantiated by the tree at the end of each day. (To be fair to the fixed clusters, we only instantiate the prefixes required to classify the day's data, rather than all possible prefixes in the clustering scheme.) Table 1 shows that the tree produced by TrackIPTree is a factor of 2.5-17 smaller with the ISP logs, and a factor of 20-100 smaller with the enterprise logs. These numbers highlight that the apriori fixed clusters are perhaps too coarse to classify accurately in parts of the IP address space, and also are insufficiently aggregated in other parts of the address space.

**Experiment 2: Changing the Maximum Leaves Allowed** Next, we explore the effect of changing $m$, the maximum number of leaves allowed to TrackIPTree. Fig. 4(c) & 4(d) show the accuracy-coverage tradeoff for TrackIPTree when $m$ ranges between 20,000-200,000 leaves for the ISP logs, and 1,000-50,000 leaves for the enterprise logs. Clearly, in both cases, the predictive accuracy increases with $m$ only until $m$ is "sufficiently large" – once $m$ is large enough to capture all the distinct subtrees in the underlying optimal IPtree, the predictive accuracy will not increase. While the actual values of $m$ are specific to our data sets, the results highlight the importance of having a space-efficient and flexible algorithm – both 10,000 and 100,000 are very modest sizes compared to the number of possible apriori fixed clusters, or the size of the IPv4 address space, and this suggests that the underlying decision tree required is indeed of a modest size.

**Experiment 3: Does a Dynamic Tree Help?** In this experiment, we demonstrate empirically that our algorithm's dynamic aspects do indeed significantly enhance its accuracy over static clustering schemes. The static clustering that we compare to is a tree generated by our algorithm, but one that learns over the first $z$ days, and then stays unchanged. For ease of reference, we call such a tree a $z$-static tree; in our experiments, we set $z = 5$ and $z = 10$. We compare these trees by examining separately the errors incurred on legitimate and spam IPs.

|  | ISP | Enterprise |
|---|---|---|
| TrackIPTree | 99942 | 9963 |
| /24 Prefixes | 1732441 | 1426445 |
| Network-aware | 260132 | 223025 |

Table 1: Sizes of Clustering Schemes

| $w_t$ | Implication | Colour |
|---|---|---|
| $\geq 0.2$ | Strongly Legit | Dark Green |
| $[0, 0.2)$ | Weakly Legit | Light Green |
| $(-0.2, 0)$ | Weakly Malicious | Blue |
| $\leq -0.2$ | Strongly Malicious | White |

Table 2: Colour coding for IPtree in Fig 1(b)

Fig. 4(e) & 4(f) compare the errors of the $z$-static trees and the dynamic tree on legitimate and spam IPs respectively, using the ISP logs. Clearly, *both $z$-static trees degrade in accuracy over time*, and they do so on both legitimate and spam IPs. On the other hand, the accuracy of the dynamic tree does not degrade over this period. Further, the in error grows with time; after 28 days, the 10-static tree has almost a factor of 2 higher error on both spam IPs and legitimate IPs.

**Discussion and Implications** Our experiments demonstrate that our algorithm is able to achieve high accuracy in predicting legitimate and spam IPs, e.g., it can predict $95\%$ of the spam IPs correctly, when misclassifying only $5\%$ of the legitimate IPs. However, it does not classify the IPs perfectly. This is unsurprising – achieving zero classification error in these applications is practically infeasible, given IP address dynamics [25]. Nevertheless, our IPtree still provides insight into the malicious activity on the Internet.

As an example, we examine a high-level view of the Internet obtained from our tree, and its implications. Fig. 1(b) visualizes an IPtree on the ISP logs with 50,000 leaves. It is laid out so that the root prefix is near the center, and the prefixes grow their children outwards. The nodes are coloured depending on their weights, as shown in Table 2: for node $t$, define $w_t = \sum_{j \in Q} x_j(y_{j,+} - y_{j,-})$, where $Q$ is the set of prefixes of node $t$ (including node $t$ itself. Thus, the blue central nodes are the large prefixes (e.g., /8 prefixes), and the classification they output is slightly malicious; this means that an IP address without a longer matching prefix in the tree is typically classified to be malicious. This suggests, for example, that an unseen IP address is typically classified as a spammer by our IPtree, which is consistent with the observations of network administrators. A second observation we can make is that the tree has many short branches as well as long branches, suggesting that some IP prefixes are grown to much greater depth than others. This might happen, for instance, if active IP addresses for this application are not distributed uniformly in the address space (and so all prefixes do not need to be grown at uniform rates), which is also what we might expect to see based on prior work [16].

Of course, these observations are only examples; a complete analysis of our IPtree's implications is part of our future work. Nevertheless, these observations suggest that our tree does indeed capture an appropriate picture of the malicious activity on the Internet.

## 6 Other Related Work

In the networking and databases literature, there has been much interest in designing streaming algorithms to identify IP prefixes with significant network traffic [7, 9, 27], but these algorithms do not explore how to predict malicious activity. Previous IP-based approaches to reduce spam traffic [22, 24], as mentioned earlier, have also explored individual IP addresses, which are not particularly useful since they are so dynamic [15, 19, 25]. Zhang et al [26] also examine how to predict whether known malicious IP addresses may appear at a given network, by analyzing the co-occurence of all known malicious IP addresses at a number of different networks. More closely related is [21], who present algorithms to extract prefix-based filtering rules for IP addresses that may be used in offline settings. There has also been work on computing decision trees over streaming data [8, 13], but this work assumes that data comes from a fixed distribution.

## 7 Conclusion

We have addressed the problem of discovering dynamic malicious regions on the Internet. We model this problem as one of adaptively pruning a known decision tree, but with the additional challenges coming from real-world settings – severe space requirements and a changing target function. We developed new algorithms to address this problem, by combining "experts" algorithms and online paging algorithms. We showed guarantees on our algorithm's performance as a function of the best possible pruning of a similar size, and our experimental results on real-world datasets are orders of magnitude better than current approaches.

**Acknowledgements** We are grateful to Alan Glasser and Gang Yao for their help with the data analysis efforts.

## Footnotes

[1]We leave the exact statement of the guarantee to the proof in [23]

[2]For space reasons, we defer the details of how we assign this labeling to the technical report [23]

# References

[1] Brightmail. http://www.brightmail.com.

[2] SpamAssassin. http://www.spamassassin.apache.org.

[3] SpamHaus. http://www.spamhaus.net.

[4] BLUM, A., AND MANSOUR, Y. From external to internal regret. In *In Proceedings of 18th Annual Conference on Computational Learning Theory (COLT 2005)* (2005).

[5] CESA-BIANCHI, N., FREUND, Y., HAUSSLER, D., HELMBOLD, D. P., SCHAPIRE, R. E., AND WAR-MUTH, M. K. How to use expert advice. *J. ACM 44*, 3 (1997), 427–485.

[6] COLLINS, M. P., SHIMEALL, T. J., FABER, S., NAIES, J., WEAVER, R., AND SHON, M. D. Using uncleanliness to predict future botnet addresses. In *Proceedings of the Internet Measurement Conference* (2007).

[7] CORMODE, G., KORN, F., MUTHUKRISHNAN, S., AND SRIVASTAVA, D. Diamond in the rough: Finding hierarchical heavy hitters in multi-dimensional data. In *SIGMOD '04: Proceedings of the 2004 ACM SIGMOD international conference on Management of data* (2004).

[8] DOMINGOS, P., AND HULTEN, G. Mining high-speed data streams. In *Proceedings of ACM SIGKDD* (2000), pp. 71–80.

[9] ESTAN, C., SAVAGE, S., AND VARGHESE, G. Automatically inferring patterns of resource consumption in network traffic. In *Proceedings of SIGCOMM'03* (2003).

[10] FREUND, Y., SCHAPIRE, R. E., SINGER, Y., AND WARMUTH, M. K. Using and combining predictors that specialize. In *Proceedings of the Twenty-Ninth Annual Symposium on the Theory of Computing (STOC)* (1997), pp. 334–343.

[11] HELMBOLD, D. P., AND SCHAPIRE, R. E. Predicting nearly as well as the best pruning of a decision tree. *Machine Learning 27*, 1 (1997), 51–68.

[12] HERBSTER, M., AND WARMUTH, M. Tracking the best expert. *Machine Learning 32*, 2 (August 1998).

[13] JIN, R., AND AGARWAL, G. Efficient and effective decision tree construction on streaming data. In *Proceedings of ACM SIGKDD* (2003).

[14] JUNG, J., KRISHNAMURTHY, B., AND RABINOVICH, M. Flash crowds and denial of service attacks: Characterization and implications for cdns and websites. In *Proceedings of the International World Wide Web Conference* (May 2002).

[15] JUNG, J., AND SIT, E. An empirical study of spam traffic and the use of DNS black lists. In *Proceedings of Internet Measurement Conference (IMC)* (2004).

[16] KOHLER, E., LI, J., PAXSON, V., AND SHENKER, S. Observed structure of addresses in IP traffic. *IEEE/ACM Transactions in Networking 14*, 6 (2006).

[17] KRISHNAMURTHY, B., AND WANG, J. On network-aware clustering of web clients. In *Proceedings of ACM SIGCOMM* (2000).

[18] MAO, Z. M., SEKAR, V., SPATSCHECK, O., VAN DER MERWE, J., AND VASUDEVAN, R. Analyzing large ddos attacks using multiple data sources. In *ACM SIGCOMM Workshop on Large Scale Attack Defense* (2006).

[19] RAMACHANDRAN, A., AND FEAMSTER, N. Understanding the network-level behavior of spammers. In *Proceedings of ACM SIGCOMM* (2006).

[20] SLEATOR, D. D., AND TARJAN, R. E. Amortized efficiency of list update and paging rules. In *Communications of the ACM* (1985), vol. 28, pp. 202–208.

[21] SOLDO, F., MARKOPOULO, A., AND ARGYRAKI, K. Optimal filtering of source address prefixes: Models and algorithms. In *Proceedings of IEEE Infocom 2009* (2009).

[22] TWINING, D., WILLIAMSON, M. M., MOWBRAY, M., AND RAHMOUNI, M. Email prioritization: Reducing delays on legitimate mail caused by junk mail. In *USENIX Annual Technical Conference* (2004).

[23] VENKATARAMAN, S., BLUM, A., SONG, D., SEN, S., AND SPATSCHECK, O. Tracking dynamic sources of malicious activity at internet-scale. Tech. Rep. TD-7NZS8K, AT&T Labs, 2009.

[24] VENKATARAMAN, S., SEN, S., SPATSCHECK, O., HAFFNER, P., AND SONG, D. Exploiting network structure for proactive spam mitigation. In *Proceedings of Usenix Security'07* (2007).

[25] XIE, Y., YU, F., ACHAN, K., GILLUM, E., , GOLDSZMIDT, M., AND WOBBER, T. How dynamic are IP addresses? In *Proceedings of ACM SIGCOMM* (2007).

[26] ZHANG, J., PORRAS, P., AND ULRICH, J. Highly predictive blacklists. In *Proceedings of Usenix Security'08* (2008).

[27] ZHANG, Y., SINGH, S., SEN, S., DUFFIELD, N., AND LUND, C. Online identification of hierarchical heavy hitters: algorithms, evaluation, and applications. In *IMC '04: Proceedings of the 4th ACM SIGCOMM conference on Internet measurement* (New York, NY, USA, 2004), ACM, pp. 101–114.

